# Fast Iterative Kernel PCA

**Nicol N. Schraudolph    Simon Günter    S.V. N. Vishwanathan**
{nic.schraudolph,simon.guenter,svn.vishwanathan}@nicta.com.au

Statistical Machine Learning, National ICT Australia
Locked Bag 8001, Canberra ACT 2601, Australia

Research School of Information Sciences & Engineering
Australian National University, Canberra ACT 0200, Australia

## Abstract

We introduce two methods to improve convergence of the Kernel Hebbian Algorithm (KHA) for iterative kernel PCA. KHA has a scalar gain parameter which is either held constant or decreased as $1/t$, leading to slow convergence. Our KHA/et algorithm accelerates KHA by incorporating the reciprocal of the current estimated eigenvalues as a gain vector. We then derive and apply Stochastic Meta-Descent (SMD) to KHA/et; this further speeds convergence by performing gain adaptation in RKHS. Experimental results for kernel PCA and spectral clustering of USPS digits as well as motion capture and image de-noising problems confirm that our methods converge substantially faster than conventional KHA.

## 1  Introduction

Principal Components Analysis (PCA) is a standard linear technique for dimensionality reduction. Given a matrix $\boldsymbol{X} \in \mathbb{R}^{n \times l}$ of $l$ centered, $n$-dimensional observations, PCA performs an eigendecomposition of the covariance matrix $\boldsymbol{Q} := \boldsymbol{X}\boldsymbol{X}^\top$. The $r \times n$ matrix $\boldsymbol{W}$ whose rows are the eigenvectors of $\boldsymbol{Q}$ associated with the $r \leq n$ largest eigenvalues minimizes the least-squares reconstruction error

$$||\boldsymbol{X} - \boldsymbol{W}^\top\boldsymbol{W}\boldsymbol{X}||_F, \tag{1}$$

where $|| \cdot ||_F$ is the Frobenius norm.

As it takes $O(n^2 l)$ time to compute $\boldsymbol{Q}$ and up to $O(n^3)$ time to eigendecompose it, PCA can be prohibitively expensive for large amounts of high-dimensional data. Iterative methods exist that do not compute $\boldsymbol{Q}$ explicitly and thereby reduce the computational cost to $O(rn)$ per iteration. One such method is Sanger's [1] *Generalized Hebbian Algorithm* (GHA), which updates $\boldsymbol{W}$ as

$$\boldsymbol{W}_{t+1} = \boldsymbol{W}_t + \eta_t[\boldsymbol{y}_t\boldsymbol{x}_t^\top - \mathrm{lt}(\boldsymbol{y}_t\boldsymbol{y}_t^\top)\boldsymbol{W}_t]. \tag{2}$$

Here $\boldsymbol{x}_t \in \mathbb{R}^n$ is the observation at time $t$, $\boldsymbol{y}_t := \boldsymbol{W}_t\boldsymbol{x}_t$, and $\mathrm{lt}(\cdot)$ makes its argument lower triangular by zeroing all elements above the diagonal. For an appropriate scalar gain $\eta_t$, $\boldsymbol{W}_t$ will generally tend to converge to the principal component solution as $t \to \infty$; though its global convergence is not proven [2].

One can do better than PCA in minimizing the reconstruction error (1) by allowing *nonlinear* projections of the data into $r$ dimensions. Unfortunately such approaches often pose difficult nonlinear optimization problems. Kernel methods [3] provide a way to incorporate nonlinearity without unduly complicating the optimization problem. Kernel PCA [4] performs an eigendecomposition on the kernel expansion of the data, an $l \times l$ matrix. To reduce the attendant $O(l^2)$ space and $O(l^3)$ time complexity, Kim et al. [2] introduced the *Kernel Hebbian Algorithm* (KHA) kernelizing GHA.

Both GHA and KHA are examples of *stochastic approximation* algorithms, whose iterative updates employ individual observations in place of — but, in the limit, approximating — statistical properties of the entire data. By interleaving their updates with the passage through the data, stochastic approximation algorithms can greatly outperform conventional methods on large, redundant data sets, even though their convergence is comparatively slow.

Both the GHA and KHA updates incorporate a scalar gain parameter $\eta_t$, which is either held fixed or annealed according to some predefined schedule. Robbins and Monro [5] established conditions on the sequence of $\eta_t$ that guarantee the convergence of many stochastic approximation algorithms; a widely used annealing schedule that obeys these conditions is $\eta_t \propto \tau/(t + \tau)$, for any $\tau > 0$.

Here we propose the inclusion of a gain vector in the KHA, which provides each estimated eigenvector with its individual gain parameter. We present two methods for setting these gains: In the KHA/et algorithm, the gain of an eigenvector is reciprocal to its estimated eigenvalue as well as the iteration number $t$ [6]. Our second method, KHA-SMD, additionally employs Schraudolph's [7] *Stochastic Meta-Descent* (SMD) technique for adaptively controlling a gain vector for stochastic gradient descent, derived and applied here in Reproducing Kernel Hilbert Space (RKHS), *cf.* [8].

The following section summarizes Kim et al.'s [2] KHA. Sections 3 and 4 describe our KHA/et and KHA-SMD algorithms, respectively. We report our experiments with these algorithms in Section 5 before concluding with a discussion.

## 2 Kernel Hebbian Algorithm (KHA and KHA/t)

Kim et al. [2] apply Sanger's [1] GHA to data mapped into a *reproducing kernel Hilbert space* (RKHS) $\mathcal{H}$ via the function $\Phi : \mathbb{R}^n \to \mathcal{H}$. $\mathcal{H}$ and $\Phi$ are implicitly defined via the kernel $k : \mathbb{R}^n \times \mathbb{R}^n \to \mathcal{H}$ with the property $\forall \boldsymbol{x}, \boldsymbol{x}' \in \mathbb{R}^n: k(\boldsymbol{x}, \boldsymbol{x}') = \langle \Phi(\boldsymbol{x}), \Phi(\boldsymbol{x}') \rangle_{\mathcal{H}}$, where $\langle \cdot, \cdot \rangle_{\mathcal{H}}$ denotes the inner product in $\mathcal{H}$. Let $\boldsymbol{\Phi}$ denote the transposed mapped data:

$$\boldsymbol{\Phi} := [\Phi(\boldsymbol{x}_1), \Phi(\boldsymbol{x}_2), \ldots \Phi(\boldsymbol{x}_l)]^\top. \tag{3}$$

This assumes a fixed set of $l$ observations whereas GHA relies on an infinite sequence of observations for convergence. Following Kim et al. [2], we use an indexing function $p : \mathbb{N} \to \mathbb{Z}_l$ which concatenates random permutations of $\mathbb{Z}_l$ to reconcile this discrepancy.

PCA, GHA, and hence KHA all assume that the data is centered. Since the mapping into feature space performed by kernel methods does not necessarily preserve such centering, we must re-center the mapped data:

$$\boldsymbol{\Phi}' := \boldsymbol{\Phi} - \boldsymbol{M}\boldsymbol{\Phi}, \tag{4}$$

where $\boldsymbol{M}$ denotes the $l \times l$ matrix with entries all equal to $1/l$. This is achieved by replacing the *kernel matrix* $\boldsymbol{K} := \boldsymbol{\Phi}\boldsymbol{\Phi}^\top$ (*i.e.,* $[\boldsymbol{K}]_{ij} := k(\boldsymbol{x}_i, \boldsymbol{x}_j)$) by its centered version

$$\boldsymbol{K}' := \boldsymbol{\Phi}'\boldsymbol{\Phi}'^\top = (\boldsymbol{\Phi} - \boldsymbol{M}\boldsymbol{\Phi})(\boldsymbol{\Phi} - \boldsymbol{M}\boldsymbol{\Phi})^\top = \boldsymbol{K} - \boldsymbol{M}\boldsymbol{K} - (\boldsymbol{M}\boldsymbol{K})^\top + \boldsymbol{M}\boldsymbol{K}\boldsymbol{M}. \tag{5}$$

Since all rows of $\boldsymbol{M}\boldsymbol{K}$ are identical (as are all elements of $\boldsymbol{M}\boldsymbol{K}\boldsymbol{M}$) we can precalculate that row in $O(l^2)$ time and store it in $O(l)$ space to efficiently implement operations with the centered kernel. The kernel centered on the training data is also used when testing the trained system on new data.

From Kernel PCA [4] it is known that the principal components must lie in the span of the centered mapped data; we can therefore express the GHA weight matrix as $\boldsymbol{W}_t = \boldsymbol{A}_t\boldsymbol{\Phi}'$, where $\boldsymbol{A}$ is an $r \times l$ matrix of expansion coefficients, and $r$ the number of principal components. The GHA weight update (2) thus becomes

$$\boldsymbol{A}_{t+1}\boldsymbol{\Phi}' = \boldsymbol{A}_t\boldsymbol{\Phi}' + \eta_t[\boldsymbol{y}_t\Phi'(\boldsymbol{x}_{p(t)})^\top - \mathrm{lt}(\boldsymbol{y}_t\boldsymbol{y}_t^\top)\boldsymbol{A}_t\boldsymbol{\Phi}'], \tag{6}$$

where

$$\boldsymbol{y}_t := \boldsymbol{W}_t\Phi'(\boldsymbol{x}_{p(t)}) = \boldsymbol{A}_t\boldsymbol{\Phi}'\Phi'(\boldsymbol{x}_{p(t)}) = \boldsymbol{A}_t\boldsymbol{k}'_{p(t)}, \tag{7}$$

using $\boldsymbol{k}'_i$ to denote the $i^{\text{th}}$ column of the centered kernel matrix $\boldsymbol{K}'$. Since we have $\Phi'(\boldsymbol{x}_i)^\top = \boldsymbol{e}_i^\top\boldsymbol{\Phi}'$, where $\boldsymbol{e}_i$ is the unit vector in direction $i$, (6) can be rewritten solely in terms of expansion coefficients as

$$\boldsymbol{A}_{t+1} = \boldsymbol{A}_t + \eta_t[\boldsymbol{y}_t\boldsymbol{e}_{p(t)}^\top - \mathrm{lt}(\boldsymbol{y}_t\boldsymbol{y}_t^\top)\boldsymbol{A}_t]. \tag{8}$$

Introducing the update coefficient matrix

$$\mathbf{\Gamma}_t := \boldsymbol{y}_t \boldsymbol{e}_{p(t)}^\top - \text{lt}(\boldsymbol{y}_t \boldsymbol{y}_t^\top) \boldsymbol{A}_t \tag{9}$$

we obtain the compact update rule

$$\boldsymbol{A}_{t+1} = \boldsymbol{A}_t + \eta_t \mathbf{\Gamma}_t. \tag{10}$$

In their experiments, Kim et al. [2] employed the KHA update (8) with a constant scalar gain, $\eta_t = \text{const}$. They also proposed letting the gain decay as $\eta_t = 1/t$ for stationary data.

# 3  Gain Decay with Reciprocal Eigenvalues (KHA/et)

Consider the term $\boldsymbol{y}_t \boldsymbol{x}_t^\top = \boldsymbol{W}_t \boldsymbol{x}_t \boldsymbol{x}_t^\top$ appearing on the right-hand side of the GHA update rule (2). At the desired solution, the rows of $\boldsymbol{W}_t$ contain the principal components, *i.e.,* the leading eigenvectors of $\boldsymbol{Q} = \boldsymbol{X}\boldsymbol{X}^\top$. The elements of $\boldsymbol{y}_t$ thus scale with the associated eigenvalues of $\boldsymbol{Q}$. Wide spreads of eigenvalues can therefore lead to *ill-conditioning*, hence slow convergence, of the GHA; the same holds for the KHA.

In our KHA/et algorithm, we counteract this problem by furnishing KHA with a *gain vector* $\boldsymbol{\eta}_t$ that provides each eigenvector estimate with its individual gain parameter. The update rule (10) thus becomes

$$\boldsymbol{A}_{t+1} = \boldsymbol{A}_t + \text{diag}(\boldsymbol{\eta}_t)\, \mathbf{\Gamma}_t, \tag{11}$$

where $\text{diag}(\cdot)$ turns a vector into a diagonal matrix. To condition KHA, we set the gain parameters proportional to the reciprocal of both the iteration number $t$ and the current estimated eigenvalue; a similar apporach was used by Chen and Chang [6] for neural network feature selection. Let $\boldsymbol{\lambda}_t$ be the vector of eigenvalues associated with the current estimate (as stored in $\boldsymbol{A}_t$) of the first $r$ eigenvectors. KHA/et sets the $i$th element of $\boldsymbol{\eta}_t$ to

$$[\boldsymbol{\eta}_t]_i = \frac{||\boldsymbol{\lambda}_t||}{[\boldsymbol{\lambda}_t]_i} \frac{l}{t+l} \eta_0, \tag{12}$$

where $\eta_0$ is a free scalar parameter, and $l$ the size of the data set. This conditions the KHA update (8) by proportionately decreasing (increasing) the gain for rows of $\boldsymbol{A}_t$ associated with large (small) eigenvalues.

The norm $||\boldsymbol{\lambda}_t||$ in the numerator of (12) is maximized by the principal components; its growth serves to counteract the $l/(t+l)$ gain decay while the leading eigenspace is idientified. This achieves an effect comparable to an adaptive "search then converge" gain schedule [9] without introducing any tuning parameters.

As the goal of KHA is to find the eigenvectors in the first place, we don't know the true eigenvalues while running the algorithm. Instead we use the eigenvalues associated with KHA's current eigenvector estimate, computed as

$$[\boldsymbol{\lambda}_t]_i = \frac{||[\boldsymbol{A}_t]_{i*}\boldsymbol{K}'||}{||[\boldsymbol{A}_t]_{i*}||} \tag{13}$$

where $[\boldsymbol{A}_t]_{i*}$ denotes the $i$-th row of $\boldsymbol{A}_t$. This can be stated compactly as

$$\boldsymbol{\lambda}_t = \sqrt{\frac{\text{diag}[\boldsymbol{A}_t\boldsymbol{K}'(\boldsymbol{A}_t\boldsymbol{K}')^\top]}{\text{diag}(\boldsymbol{A}_t\boldsymbol{A}_t^\top)}} \tag{14}$$

where the division and square root operation are performed element-wise, and $\text{diag}(\cdot)$ (when applied to a matrix) extracts the vector of elements along the matrix diagonal.

Note that naive computation of $\boldsymbol{A}\boldsymbol{K}'$ is quite expensive: $O(rl^2)$. Since the eigenvalues evolve gradually, it suffices to re-estimate them only occasionally; we determine $\boldsymbol{\lambda}_t$ and $\boldsymbol{\eta}_t$ once for each pass through the training data set, *i.e.,* every $l$ iterations. Below we derive a way to maintain $\boldsymbol{A}\boldsymbol{K}'$ incrementally in an affordable $O(rl)$ via Equations (17) and (18).

# 4 KHA with Stochastic Meta-Descent (KHA-SMD)

While KHA/et makes reasonable assumptions about how the gains of a KHA update should be scaled, it is by no means clear how close the resulting gains are to being optimal. To explore this question, we now derive and implement the *Stochastic Meta-Descent* (SMD [7]) algorithm for KHA/et. SMD controls gains adaptively in response to the observed history of parameter updates so as to optimize convergence. Here we focus on the specifics of applying SMD to KHA/et; please refer to [7, 8] for more general derivations and discussion of SMD.

Using the KHA/et gains as a starting point, the KHA-SMD update is

$$\boldsymbol{A}_{t+1} = \boldsymbol{A}_t + e^{\operatorname{diag}(\boldsymbol{\rho}_t)} \operatorname{diag}(\boldsymbol{\eta}_t) \, \boldsymbol{\Gamma}_t, \tag{15}$$

where the log-gain vector $\boldsymbol{\rho}_t$ is adjusted by SMD. (Note that the exponential of a diagonal matrix is obtained simply by exponentiating the individual diagonal entries.)

In an RKHS, SMD adapts a *scalar* log-gain whose update is driven by the inner product between the gradient and a differential of the system parameters, all in the RKHS [8]. Note that $\boldsymbol{\Gamma}_t \boldsymbol{\Phi}'$ can be interpreted as the gradient in the RKHS of the (unknown) merit function maximized by KHA, and that (15) can be viewed as $r$ coupled updates in RKHS, one for each row of $\boldsymbol{A}_t$, each associated with a scalar gain. SMD-KHA's adaptation of the log-gain vector is therefore driven by the diagonal entries of $\langle \boldsymbol{\Gamma}_t \boldsymbol{\Phi}', \boldsymbol{B}_t \boldsymbol{\Phi}' \rangle_{\mathcal{H}}$, where $\boldsymbol{B}_t := \mathrm{d}\boldsymbol{A}_t$ denotes the $r \times l$ matrix of expansion coefficients for SMD's differential parameters:

$$\begin{aligned} \boldsymbol{\rho}_t &= \boldsymbol{\rho}_{t-1} + \mu \operatorname{diag}(\langle \boldsymbol{\Gamma}_t \boldsymbol{\Phi}', \boldsymbol{B}_t \boldsymbol{\Phi}' \rangle_{\mathcal{H}}) \\ &= \boldsymbol{\rho}_{t-1} + \mu \operatorname{diag}(\boldsymbol{\Gamma}_t \boldsymbol{\Phi}' \boldsymbol{\Phi}'^{\top} \boldsymbol{B}_t^{\top}) = \boldsymbol{\rho}_{t-1} + \mu \operatorname{diag}(\boldsymbol{\Gamma}_t \boldsymbol{K}' \boldsymbol{B}_t^{\top}), \end{aligned} \tag{16}$$

where $\mu$ is a scalar tuning parameter. Naive computation of $\boldsymbol{\Gamma}_t \boldsymbol{K}'$ in (16) would cost $O(rl^2)$ time, which is prohibitively expensive for large $l$. We can, however, reduce this cost to $O(rl)$ by noting that (9) implies

$$\boldsymbol{\Gamma}_t \boldsymbol{K}' = \boldsymbol{y}_t \boldsymbol{e}_{p(t)}^{\top} \boldsymbol{K}' - \operatorname{lt}(\boldsymbol{y}_t \boldsymbol{y}_t^{\top}) \boldsymbol{A}_t \boldsymbol{K}' = \boldsymbol{y}_t \boldsymbol{k}_{p(t)}'^{\top} - \operatorname{lt}(\boldsymbol{y}_t \boldsymbol{y}_t^{\top}) \boldsymbol{A}_t \boldsymbol{K}', \tag{17}$$

where the $r \times l$ matrix $\boldsymbol{A}_t \boldsymbol{K}'$ can be stored and updated incrementally via (15):

$$\boldsymbol{A}_{t+1} \boldsymbol{K}' = \boldsymbol{A}_t \boldsymbol{K}' + e^{\operatorname{diag}(\boldsymbol{\rho}_t)} \operatorname{diag}(\boldsymbol{\eta}_t) \, \boldsymbol{\Gamma}_t \boldsymbol{K}'. \tag{18}$$

The initial computation of $\boldsymbol{A}_1 \boldsymbol{K}'$ still costs $O(rl^2)$ in general but is affordable as it is performed only once. Alternatively, the time complexity of this step can easily be reduced to $O(rl)$ by making $\boldsymbol{A}_1$ suitably sparse.

Finally, we apply SMD's standard update of the differential parameters:

$$\boldsymbol{B}_{t+1} = \xi \boldsymbol{B}_t + e^{\operatorname{diag}(\boldsymbol{\rho}_t)} \operatorname{diag}(\boldsymbol{\eta}_t) (\boldsymbol{\Gamma}_t + \xi \mathrm{d}\boldsymbol{\Gamma}_t), \tag{19}$$

where the decay factor $0 \leq \xi \leq 1$ is another scalar tuning parameter. The differential $\mathrm{d}\boldsymbol{\Gamma}_t$ of the gradient is easily computed by routine application of the rules of calculus:

$$\begin{aligned} \mathrm{d}\boldsymbol{\Gamma}_t &= \mathrm{d}[\boldsymbol{y}_t \boldsymbol{e}_{p(t)}^{\top} - \operatorname{lt}(\boldsymbol{y}_t \boldsymbol{y}_t^{\top}) \boldsymbol{A}_t] \\ &= (\mathrm{d}\boldsymbol{A}_t) \boldsymbol{k}_{p(t)}' \boldsymbol{e}_{p(t)}^{\top} - \operatorname{lt}(\boldsymbol{y}_t \boldsymbol{y}_t^{\top})(\mathrm{d}\boldsymbol{A}_t) - [\mathrm{d}\operatorname{lt}(\boldsymbol{y}_t \boldsymbol{y}_t^{\top})] \boldsymbol{A}_t \\ &= \boldsymbol{B}_t \boldsymbol{k}_{p(t)}' \boldsymbol{e}_{p(t)}^{\top} - \operatorname{lt}(\boldsymbol{y}_t \boldsymbol{y}_t^{\top}) \boldsymbol{B}_t - \operatorname{lt}(\boldsymbol{B}_t \boldsymbol{k}_{p(t)}' \boldsymbol{y}_t^{\top} + \boldsymbol{y}_t \boldsymbol{k}_{p(t)}'^{\top} \boldsymbol{B}_t^{\top}) \boldsymbol{A}_t. \end{aligned} \tag{20}$$

Inserting (9) and (20) into (19) yields the update rule

$$\begin{aligned} \boldsymbol{B}_{t+1} = \ & \xi \boldsymbol{B}_t + e^{\operatorname{diag}(\boldsymbol{\rho}_t)} \operatorname{diag}(\boldsymbol{\eta}_t) [(\boldsymbol{A}_t + \xi \boldsymbol{B}_t) \, \boldsymbol{k}_{p(t)}' \boldsymbol{e}_{p(t)}^{\top} \\ & - \operatorname{lt}(\boldsymbol{y}_t \boldsymbol{y}_t^{\top})(\boldsymbol{A}_t + \xi \boldsymbol{B}_t) - \xi \operatorname{lt}(\boldsymbol{B}_t \boldsymbol{k}_{p(t)}' \boldsymbol{y}_t^{\top} + \boldsymbol{y}_t \boldsymbol{k}_{p(t)}'^{\top} \boldsymbol{B}_t^{\top}) \boldsymbol{A}_t]. \end{aligned} \tag{21}$$

In summary, the application of SMD to KHA/et comprises Equations (16), (21), and (15), in that order. The complete KHA-SMD algorithm is given as Algorithm 1. We initialize $\boldsymbol{A}_1$ to an isotropic normal density with suitably small variance, $\boldsymbol{B}_1$ to all zeroes, and $\boldsymbol{\rho}_0$ to all ones. The worst-case time complexity of non-trivial initialization steps is given explicitly; all steps in the repeat loop have a time complexity of $O(rl)$ or less.

---

**Algorithm 1** KHA-SMD

---

1. Initialize:
    (a) calculate $\boldsymbol{MK}$, $\boldsymbol{MKM}$ — $O(l^2)$
    (b) $\boldsymbol{\rho}_0 := [1 \ldots 1]^\top$
    (c) $\boldsymbol{B}_1 := \boldsymbol{0}$
    (d) $\boldsymbol{A}_1 \sim N(\boldsymbol{0}, (rl)^{-1}\boldsymbol{I})$
    (e) calculate $\boldsymbol{A}_1\boldsymbol{K}'$ — $O(rl^2)$
2. **Repeat** for $t = 1, 2, \ldots$
    (a) calculate $\boldsymbol{\lambda}_t$ (13)
    (b) calculate $\boldsymbol{\eta}_t$ (11)
    (c) select observation $\boldsymbol{x}_{p(t)}$
    (d) calculate $\boldsymbol{y}_t$ (7)
    (e) calculate $\boldsymbol{\Gamma}_t$ (9)
    (f) calculate $\boldsymbol{\Gamma}_t\boldsymbol{K}'$ (17)
    (g) update $\boldsymbol{\rho}_{t-1} \rightarrow \boldsymbol{\rho}_t$ (16)
    (h) update $\boldsymbol{B}_t \rightarrow \boldsymbol{B}_{t+1}$ (21)
    (i) update $\boldsymbol{A}_t \rightarrow \boldsymbol{A}_{t+1}$ (15)
    (j) update $\boldsymbol{A}_t\boldsymbol{K}' \rightarrow \boldsymbol{A}_{t+1}\boldsymbol{K}'$ (18)

---

## 5 Experiments

We compared our KHA/et and KHA-SMD algorithms with KHA using either a fixed gain ($\eta_t = \eta_0$) or a scheduled gain decay ($\eta_t = \eta_0 \, l/(t + l)$, denoted KHA/t) in a number of different settings: Performing kernel PCA and spectral clustering on the well-known USPS dataset [10], replicating an image denoising experiment of Kim et al. [2], and denoising human motion capture data.

In all experiments the Kernel Hebian Algorithm (KHA) and our enhanced variants are used to find the first $r$ eigenvectors of the centered Kernel matrix $\boldsymbol{K}'$. To assess the quality of the result, we reconstruct the Kernel matrix from the found eigenvectors and measure the reconstruction error

$$\mathcal{E}(\boldsymbol{A}) := ||\boldsymbol{K}' - (\boldsymbol{A}\boldsymbol{K}')^\top \boldsymbol{A}\boldsymbol{K}'||_F, \tag{22}$$

where $||\cdot||_F$ is the Frobenius norm. The minimal reconstruction error from $r$ eigenvectors, $\mathcal{E}_{\min} := \min_{\boldsymbol{A}} \mathcal{E}(\boldsymbol{A})$, can be calculated by an eigendecomposition. This allows us to report reconstruction errors as excess errors relative to the optimal reconstruction, *i.e.*, $\mathcal{E}(\boldsymbol{A})/\mathcal{E}_{\min} - 1$.

To compare algorithms we plot the excess reconstruction error on a logarithmic scale after each pass through the entire data set. This is a fair comparison since the overhead for KHA/et and KHA-SMD is negligible compared to the time required by the KHA base algorithm. The most expensive operation, the calculation of a row of the Kernel matrix, is shared by all algorithms.

We manually tuned $\eta_0$ for KHA, KHA/t, and KHA/et; for KHA-SMD we hand-tuned $\mu$, used the same $\eta_0$ as KHA/et, and the value $\xi = 0.99$ (set *a priori*) throughout. Thus a comparable amount of tuning effort went into each algorithm. Parameters were tuned by a local search over values in the set $\{a \cdot 10^b : a \in \{1, 2, 5\}, b \in \mathbb{Z}\}$.

### 5.1 USPS Digits

Our first set of experiments was performed on a subset of the well-known USPS dataset [10], namely the first 100 samples of each digit in the USPS training data. KHA with both a dot-product kernel and a Gaussian kernel with $\sigma = 8$ [1] was used to extract the first 16 eigenvectors. The results are shown in Figure 1. KHA/et clearly outperforms KHA/t for both kernels, and KHA-SMD is able to increase the convergence speed even further.

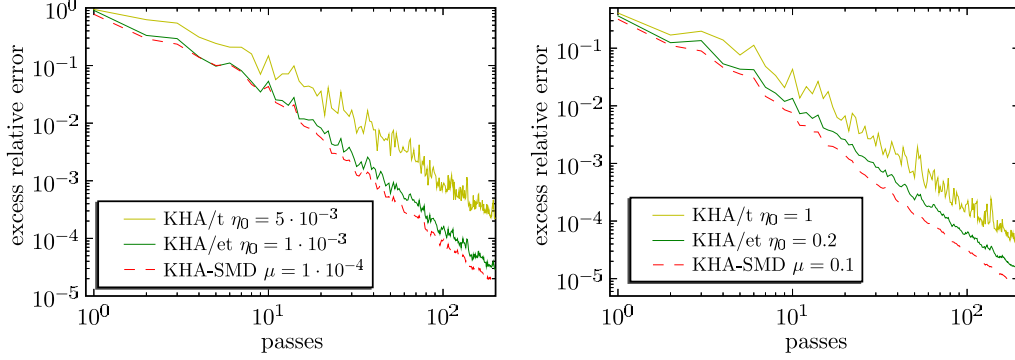

Figure 1: Excess relative reconstruction error for kernel PCA (16 eigenvectors) on USPS data, using a dot-product (left) *vs.* Gaussian kernel with $\sigma = 8$ (right).

## 5.2 Multipatch Image PCA

For our second set of experiments we replicated the image de-noising problem used by Kim et al. [2], the idea being that reconstructing image patches from their $r$ leading eigenvectors will eliminate most of the noise. The image considered here is the famous Lena picture [12] which was divided in four sub-images. From each sub-image $11 \times 11$ pixel windows were sampled on a grid with two-pixel spacing to produce 3844 vectors of 121 pixel intensity values each. The KHA with Gaussian kernel ($\sigma = 1$) was used to find the 20 best eigenvectors for each sub-image. Results averaged over all four sub-images are shown in Figure 2 (left), including KHA with the constant gain of $\eta_0 = 0.05$ employed by Kim et al. [2] for comparison.

After 50 passes through the training data, KHA/et achieves an excess reconstruction error two orders of magnitude better than conventional KHA; KHA-SMD yields an additional order of magnitude improvement. KHA/t, while superior to a constant gain, is comparatively ineffective here.

Kim et al. [2] performed 800 passes through the training data. Replicating this approach we obtain a reconstruction error of 5.64%, significantly worse than KHA/et and KHA-SMD after 50 passes. The signal-to-noise ratio (SNR) of the reconstruction after 800 passes with constant gain is 13.46 [2] while KHA/et achieves comparable performance much faster, reaching an SNR of 13.49 in 50 passes.

## 5.3 Spectral Clustering

Spectral Clustering [13] is a clustering method which includes the extraction of the first kernel PCs. In this section we present results of the spectral clustering of all 7291 patterns of the USPS data [10] where 10 kernel PCs were obtained by KHA. We used the spectral clustering method presented in

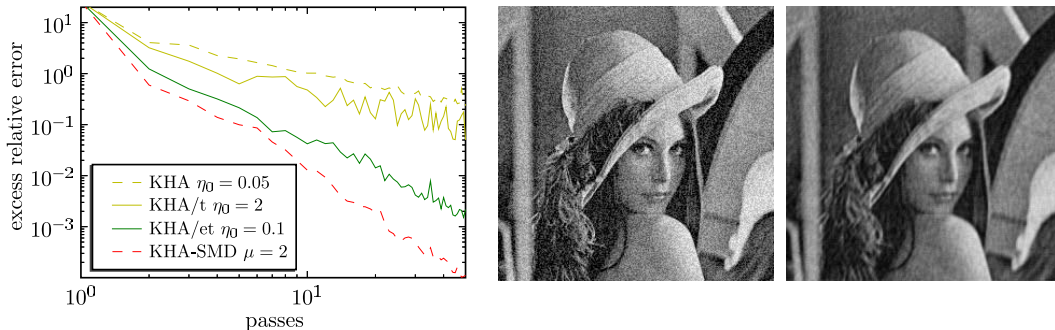

Figure 2: Excess relative reconstruction error (left) for multipatch image PCA on a noisy Lena image (center), using a Gaussian kernel with $\sigma = 1$; denoised image obtained by KHA-SMD (right).

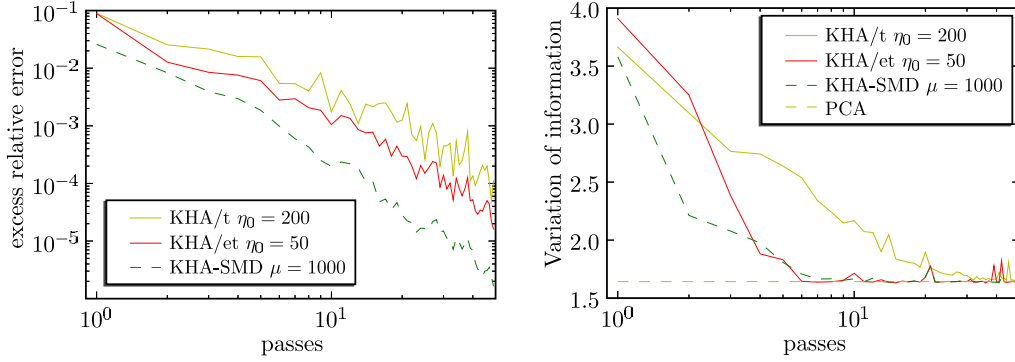

Figure 3: Excess relative reconstruction error (left) and quality of clustering as measured by variation of information (right) for spectral clustering of the USPS data with a Gaussian kernel ($\sigma = 8$).

[13], and evaluate our results via the Variation of Information (VI) metric [14], which compares the clustering obtained by spectral clustering to that induced by the class labels. On the USPS data, a VI of 4.54 corresponds to random performance, while clustering in perfect accordance with the class labels would give a VI of zero.

Our results are shown in Figure 3. Again KHA-SMD dominates KHA/et in both convergence speed and quality of reconstruction (left); KHA/et in turn outperforms KHA/t. The quality of the resulting clustering (right) reflects the quality of reconstruction. KHA/et and KHA-SMD produce a clustering as good as that obtained from a (computationally expensive) full kernel PCA within 10 passes through the data; KHA/t after more than 30 passes.

### 5.4 Human motion denoising

In our final set of experiments we employed KHA to denoise a human walking motion trajectory from the CMU motion capture database (`http://mocap.cs.cmu.edu`), converted to Cartesian coordinates via Neil Lawrence's Matlab Motion Capture Toolbox (`http://www.dcs.shef.ac.uk/~neil/mocap/`). The experimental setup was similar to that of Tangkuampien and Suter [15]: Gaussian noise was added to the frames of the original motion, then KHA with 25 PCs was used to denoise them. The results are shown in Figure 4.

As in the other experiments, KHA-SMD clearly outperformed KHA/et, which in turn was better than KHA/t. KHA-SMD managed to reduce the mean-squared error by 87.5%; it is hard to visually

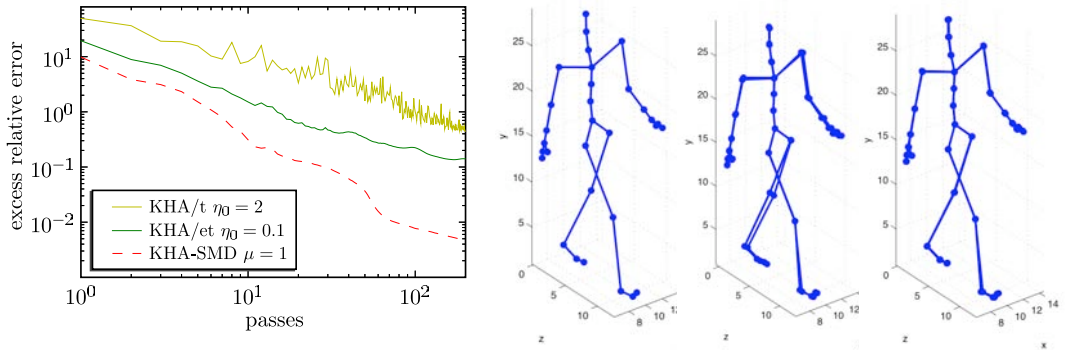

Figure 4: From left to right: Excess relative reconstruction error on human motion capture data with Gaussian kernel ($\sigma = \sqrt{1.5}$), one frame of the original data, a superposition of this original and the noisy data, and a superposition of the original and reconstructed (denoised) data.

detect a difference between the denoised frames and the original ones — see Figure 4 (right) for an example. We include movies of the original, noisy, and denoised walk in the supporting material.

# 6 Discussion

We modified Kim et al.'s [2] Kernel Hebbian Algorithm (KHA) by providing a separate gain for each eigenvector estimate. We then presented two methods, KHA/et and KHA-SMD, to set those gains. KHA/et sets them inversely proportional to the estimated eigenvalues and iteration number; KHA-SMD enhances that further by applying Stochastic Meta-Descent (SMD [7]) to perform gain adaptation in RKHS [8]. In four different experimental settings both methods were compared to a conventional gain decay schedule. As measured by relative reconstruction error, KHA-SMD clearly outperformed KHA/et, which in turn outperformed the scheduled decay, in all our experiments.

**Acknowledgments**

National ICT Australia is funded by the Australian Government's Department of Communications, Information Technology and the Arts and the Australian Research Council through Backing Australia's Ability and the ICT Center of Excellence program. This work is supported by the IST Program of the European Community, under the Pascal Network of Excellence, IST-2002-506778.

## Footnotes

[1] This is the value of $\sigma$ used by Mika et al. [11].

[2]Kim et al. [2] reported an SNR of 14.09; the discrepancy is due to different reconstruction methods.

# References

[1] T. D. Sanger. Optimal unsupervised learning in a single-layer linear feedforward network. *Neural Networks*, 2:459–473, 1989.

[2] K. I. Kim, M. O. Franz, and B. Schölkopf. Iterative kernel principal component analysis for image modeling. *IEEE Transactions on Pattern Analysis and Machine Intelligence*, 27(9): 1351–1366, 2005.

[3] B. Schölkopf and A. Smola. *Learning with Kernels*. MIT Press, Cambridge, MA, 2002.

[4] B. Schölkopf, A. J. Smola, and K.-R. Müller. Nonlinear component analysis as a kernel eigenvalue problem. *Neural Computation*, 10:1299–1319, 1998.

[5] H. Robbins and S. Monro. A stochastic approximation method. *Annals of Mathematical Statistics*, 22:400–407, 1951.

[6] L.-H. Chen and S. Chang. An adaptive learning algorithm for principal component analysis. *IEEE Transaction on Neural Networks*, 6(5):1255–1263, 1995.

[7] N. N. Schraudolph. Fast curvature matrix-vector products for second-order gradient descent. *Neural Computation*, 14(7):1723–1738, 2002.

[8] S. V. N. Vishwanathan, N. N. Schraudolph, and A. J. Smola. Step size adaptation in reproducing kernel Hilbert space. *Journal of Machine Learning Research*, 7:1107–1133, 2006.

[9] C. Darken and J. E. Moody. Towards faster stochastic gradient search. In J. E. Moody, S. J. Hanson, and R. Lippmann, editors, *Advances in Neural Information Processing Systems 4*, pages 1009–1016. Morgan Kaufmann Publishers, 1992.

[10] Y. LeCun, B. Boser, J. S. Denker, D. Henderson, R. E. Howard, W. Hubbard, and L. J. Jackel. Backpropagation applied to handwritten zip code recognition. *Neural Computation*, 1:541–551, 1989.

[11] S. Mika, B. Schölkopf, A. J. Smola, K.-R. Müller, M. Scholz, and G. Rätsch. Kernel PCA and de-noising in feature spaces. In M. S. Kearns, S. A. Solla, and D. A. Cohn, editors, *Advances in Neural Information Processing Systems 11*, pages 536–542. MIT Press, 1999.

[12] D. J. Munson. A note on Lena. *IEEE Trans. Image Processing*, 5(1), 1996.

[13] A. Ng, M. Jordan, and Y. Weiss. Spectral clustering: Analysis and an algorithm (with appendix). In T. G. Dietterich, S. Becker, and Z. Ghahramani, editors, *Advances in Neural Information Processing Systems 14*, 2002.

[14] M. Meila. Comparing clusterings: an axiomatic view. In *ICML '05: Proceedings of the 22nd international conference on Machine learning*, pages 577–584, New York, NY, USA, 2005. ACM Press.

[15] T. Tangkuampien and D. Suter. Human motion de-noising via greedy kernel principal component analysis filtering. In *Proc. Intl. Conf. Pattern Recognition*, 2006.
